# Cooled and Relaxed Survey Propagation for MRFs

**Hai Leong Chieu**[1,2]**, Wee Sun Lee**[2]
[1]Singapore MIT Alliance
[2]Department of Computer Science
National University of Singapore
haileong@nus.edu.sg,leews@comp.nus.edu.sg

**Yee-Whye Teh**
Gatsby Computational Neuroscience Unit
University College London
ywteh@gatsby.ucl.ac.uk

## Abstract

We describe a new algorithm, Relaxed Survey Propagation (RSP), for finding MAP configurations in Markov random fields. We compare its performance with state-of-the-art algorithms including the max-product belief propagation, its sequential tree-reweighted variant, residual (sum-product) belief propagation, and tree-structured expectation propagation. We show that it outperforms all approaches for Ising models with mixed couplings, as well as on a web person disambiguation task formulated as a supervised clustering problem.

## 1 Introduction

Energy minimization is the problem of finding a maximum a posteriori (MAP) configuration in a Markov random field (MRF). A MAP configuration is an assignment of values to variables that maximizes the probability (or minimizes the energy) in the MRF. Energy minimization has many applications; for example, in computer vision it is used for applications such as stereo matching [11]. As energy minimization in general MRFs is computationally intractable, approximate inference algorithms based on belief propagation are often necessary in practice. Such algorithms are split into two classes: max-product and variants address the problem by trying to find a MAP configuration directly, while sum-product and variants return approximate marginal distributions which can be used to estimate a MAP configuration. It has been shown that the max-product algorithm converges to neighborhood optimums [18], while the sum-product algorithm converges to local minima of the Bethe approximation [20]. Convergence of these algorithms are important for good approximations. Recent work (e.g. [16, 8]) on sufficient conditions for convergence of sum-product algorithms suggests that they converge better on MRFs containing potentials with small strengths. In this paper, we propose a novel algorithm, called Relaxed Survey Propagation (RSP), based on performing the sum-product algorithm on a relaxed MRF. In the relaxed MRF, there is a parameter vector y that can be optimized for convergence. By optimizing y to reduce the strengths of potentials, we show empirically that RSP converges on MRFs where other algorithms fail to converge.

The relaxed MRF is built in two steps, by first (i) converting the energy minimization problem into its weighted MAX-SAT equivalent [17], and then (ii) constructing a relaxed version of the survey propagation MRF proposed in [14]. We prove that the relaxed MRF has approximately equal joint distribution (and hence the same MAP and marginals) as the original MRF, independent (to a large extent) of the parameter vector y. Empirically, we show that RSP, when run at low temperatures ("cooled"), performs well for the application of energy minimization. For max-product algorithms, we compare against the max-product algorithm and its sequential tree-reweighted variant, which is guaranteed to converge [11]. For sum-product algorithms, we compare against residual belief propagation [6] as a state-of-the-art asynchronous belief propagation algorithm, as well as the tree-structured expectation propagation [15], which has been shown to be a special case of generalized belief propagation [19]. We show that RSP outperforms all approaches for Ising models with mixed couplings, as well as in a supervised clustering application for web person disambiguation.

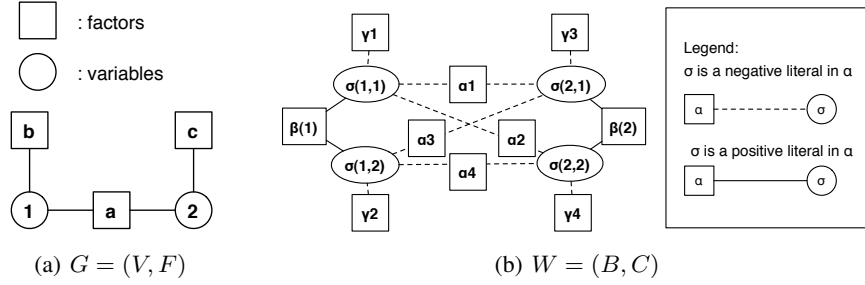

(a) $G = (V, F)$     (b) $W = (B, C)$

Figure 1: The variables $x_1$, $x_2$ in (a) are binary, resulting in 4 variables in (b). The clauses $\alpha_1$ to $\alpha_4$ in (b) are entries in the factor $a$ in (a), $\gamma_1$ and $\gamma_2$ (resp. $\gamma_3$ and $\gamma_4$) are from $b$ (resp. $c$). $\beta(1)$ and $\beta(2)$ are the positivity clauses. The *relaxed* MRF for RSP has a similar form to the graph in (b).

## 2   Preliminaries

A MRF, $G = (V, F)$, is defined by a set of variables $V$, and a set of factors $F = \{\Phi_a\}$, where each $\Phi_a$ is a non-negative function depending on $\mathbf{X_a} \subseteq V$. We assume for simplicity that variables in $V$ have the same cardinality $q$, taking values in $Q = \{1, .., q\}$. For $X_i \in V$ and $\mathbf{X_a} \subseteq V$, we denote by $x_i$ the event that $X_i = x_i$, and by $\mathbf{x_a}$ the event $\mathbf{X_a} = \mathbf{x_a}$. To simplify notation, we will sometimes write $i \in V$ for $X_i \in V$, or $a \in F$ for $\Phi_a \in F$. The joint distribution over configurations is defined by $P(\mathbf{x}) = \frac{1}{Z} \prod_a \Phi_a(\mathbf{x_a})$ where $Z$ is the normalization factor. When each $\Phi_a$ is a positive function, the joint distribution can be written as $P(\mathbf{x}) = \frac{1}{Z} \exp(-E(\mathbf{x})/T)$ where $E(\mathbf{x}) = \sum_a -\log \Phi_a(\mathbf{x_a})$ is the energy function, and the temperature $T$ is set to 1. A factor graph [13] is a graphical representation of a MRF, in the form of a bipartite graph with two types of nodes, the variables and the factors. Each factor $\Phi_a$ is connected to the variables in $\mathbf{X_a}$, and each variable $X_i$ is connected to the set of factors, $N(i)$, that depends on it. See Figure 1(a) for a simple example.

**Weighted MAX-SAT conversion [17]:**  Before describing RSP, we describe the weighted MAX-SAT (WMS) conversion of the energy minimization problem for a MRF. The WMS problem is a generalization of the satisfiability problem (SAT). In SAT, a set of boolean variables are constrained by a boolean function in conjunctive normal form, which can be treated as a set of clauses. Each clause is a set of literals (a variable or its negation), and is satisfied if one of its literals evaluates to 1. The SAT problem consist of finding a configuration that satisfies all the clauses. In WMS, each clause has a weight, and the WMS problem consists of finding a configuration with maximum total weight of satisfied clauses (called the weight of the configuration). We describe the conversion [17] of a MRF $G = (V, F)$ into a WMS problem $W = (B, C)$, where $B$ is the set of boolean variables and $C$ the set of weighted clauses. Without lost of generality, we normalize factors in $F$ to take values in $(0, 1]$. For each $X_i \in V$, introduce the variables $\sigma_{(i,x_i)} \in B$ as the predicate that $X_i = x_i$. For convenience, we index variables in $B$ either by $k$ or by $(i, x_i)$, denote factors in $F$ with Roman alphabet (e.g. $a, b, c$) and clauses in $C$ with Greek alphabet (e.g. $\alpha, \beta, \gamma$). For a clause $\alpha \in C$, we denote by $C(\alpha)$ as the set of variables in $\alpha$. There are two types of clauses in $C$: *interaction* and *positivity* clauses.

**Definition 1.** *Interaction clauses: For each entry $\Phi_a(\mathbf{x_a})$ in $\Phi_a \in F$, introduce the clause $\alpha = \vee_{x_i \in \mathbf{x_a}} \overline{\sigma_{(i,x_i)}}$ with the weight $w_\alpha = -\log(\Phi_a(\mathbf{x_a}))$. We write $\alpha \sqsubset a$ to show that the clause $\alpha$ comes from the factor $\Phi_a \in F$, and we denote $a = \mathrm{src}(\alpha)$ to be the factor $\Phi_a \in F$ for which $\alpha \sqsubset a$.*

The violation of an interaction clause corresponding to $\Phi_a(\mathbf{x_a})$ entails that $\sigma_{(i,x_i)} = 1$ for all $x_i \in \mathbf{x_a}$. This correspond to the event that $X_i = x_i$ for $X_i \in \mathbf{X_a}$.

**Definition 2.** *Positivity clauses: for $X_i \in V$, introduce the clause $\beta(i) = \vee_{x_i \in Q} \sigma_{(i,x_i)}$ with weights $w_{\beta(i)} = 2 * \sum_{\alpha \in C_i} w_\alpha$, where $C_i$ is the set of interaction clauses containing any variable in $\{\sigma_{(i,x_i)}\}_{x_i \in Q}$. For $X_i \in V$, we denote $\beta(i)$ as the corresponding positivity clause in $C$, and for a positivity clause $\beta \in C$, we denote $\mathrm{src}(\beta)$ for the corresponding variable in $V$.*

Positivity clauses have large weights to ensure that for each $X_i \in V$, at least one predicate in $\{\sigma_{(i,x_i)}\}_{x_i \in Q}$ equals 1. To map $\sigma$ back to a configuration in the original MRF, exactly one variable in each set of $\{\sigma_{(i,x_i)}\}_{x_i \in Q}$ can take the value 1. We call such configurations valid configurations:

**Definition 3.** *A configuration is valid if $\forall X_i \in V$, exactly one of the indicators $\{\sigma_{i,x_i}\}_{x_i \in Q}$ equals 1. There are two types of invalid configurations:* **MTO** *configurations where more than one variable in the set $\{\sigma_{i,x_i}\}_{x_i \in Q}$ equals 1, and* **AZ** *configurations where all variables in the set equals zero . For valid configurations $\sigma$, let $\mathbf{x}(\sigma)$ be the corresponding configuration of $\sigma$ in $V$.*

For valid configurations $\sigma$, and for each $a \in F$, exactly one interaction clause in $\{\alpha\}_{\alpha \sqsubset a}$ is violated: when $\alpha$ corresponding to $\Phi_a(\mathbf{x_a})$ is violated, we have $\mathbf{X_a} = \mathbf{x_a}$ in $\mathbf{x}(\sigma)$. Valid configurations have locally maximal weights [17]: MTO configurations have low weights since in all interaction clauses, variables appear as negative literals. AZ configurations have low weights because they violate the positivity clauses. See Figure 1 for an example of a WMS equivalent of a simple factor graph.

## 3 Relaxed Survey Propagation

In this section, we transform the WMS problem $W = (B, C)$ into another MRF, $G_s = (V_s, F_s)$, based on the construction of the MRF for survey propagation [14]. We show (in Section 3.1) that under this framework, we are able to remove MTO configurations, and AZ configurations have negligible probability. In survey propagation, in addition to the values $\{0, 1\}$, variables can take a third value, * ("joker" state), signifying that the variable is free to take either 0 or 1, without violating any clause. In this section, we assume that variables $\sigma_k$ take values in $\{0, 1, *\}$.

**Definition 4.** *[14] A variable $\sigma_k$ is constrained by the clause $\alpha \in C$ if it is the unique satisfying variable for clause $\alpha$ (all other variables violate $\alpha$). Define $\mathrm{CON}_{k,\alpha}(\sigma_{C(\alpha)}) = \delta(\sigma_k$ is constrained by $\alpha)$, where $\delta(P)$ equals 1 if the predicate $P$ is true, and 0 otherwise.*

We introduce the parameters $\{y_a\}_{a \in F}$ and $\{y_i\}_{i \in V}$ by modifying the definition of VAL in [14]:

**Definition 5.** *An assignment $\sigma$ is invalid for clause $\alpha$ if and only if all variables are unsatisfying except for exactly one for which $\sigma_k = *$. (In this case, $\sigma_k$ cannot take * as it is constrained). Define*

$$\mathrm{VAL}_\alpha(\sigma_{C(\alpha)}) = \begin{cases} \exp(w_\alpha) & \textit{if } \sigma_{C(a)} \textit{ satisfies } \alpha \\ \exp(-y_{\mathrm{src}(\alpha)}) & \textit{if } \sigma_{C(a)} \textit{ violates } \alpha \\ 0 & \textit{if } \sigma_{C(a)} \textit{ is invalid} \end{cases} \tag{1}$$

The term $\exp(-y_{\mathrm{src}(\alpha)})$ is the penalty for violating clauses, with $\mathrm{src}(\alpha) \in V \cup F$ defined in Definitions 1 and 2. For interaction clauses, we index $y_a$ by $a \in F$ because among valid configurations, exactly one clause in the group $\{\alpha\}_{\alpha \sqsubset a}$ is violated, and $\exp(-y_a)$ becomes a constant factor. Positivity clauses are always satisfied and the penalty factor will not appear for valid configurations.

**Definition 6.** *[14] Define the parent set $P_i$ of a variable $\sigma_k$ to be the set of clauses for which $\sigma_k$ is the unique satisfying variable, (i.e. the set of clauses constraining $\sigma_k$).*

We now construct the MRF $G_s = (V_s, F_s)$ where variables $\lambda_k \in V_s$ are of the form $\lambda_k = (\sigma_k, P_k)$, with $\sigma_k$ variables in the WMS problem $W = (B, C)$. (See Figure 1). We define single-variable compatibilities ($\Psi_k$) and clause compatibilities ($\Psi_\alpha$) as in [14]:

$$\Psi_k(\lambda_k = \{\sigma_k, P_k\}) = \begin{cases} \omega_0 \text{ if } P_k = \emptyset, \sigma_k \neq * \\ \omega_* \text{ if } P_k = \emptyset, \sigma_k = * \\ 1 \text{ for any other valid } (\sigma_k, P_k) \end{cases}, \text{ where } \omega_0 + \omega_* = 1 \tag{2}$$

$$\Psi_\alpha(\lambda_\alpha = \{\sigma_k, P_k\}_{k \in C(\alpha)}) = \mathrm{VAL}_\alpha(\sigma_{C(\alpha)}) \times \prod_{k \in C(\alpha)} \delta((\alpha \in P_k) = \mathrm{CON}_{\alpha,k}(\sigma_{C(\alpha)})), \tag{3}$$

where $\delta$ is defined in Definition 4. The single-variable compatibilities $\Psi_k(\lambda_k)$ are defined so that when $\sigma_k$ is unconstrained (i.e. $P_k = \emptyset$), $\Psi_k(\lambda_k)$ takes the values $\omega_*$ or $\omega_0$ depending on whether $\sigma_k$ equals *. The clause compatibilities introduce the clause weights and penalties into the joint distribution. The factor graph has the following underlying distribution:

$$P(\{\sigma_k, P_k\}_k) \propto \omega_0^{n_0} \omega_*^{n_*} \prod_{\alpha \in \mathrm{SAT}(\sigma)} \exp(w_\alpha) \prod_{\alpha \in \mathrm{UNSAT}(\sigma)} \exp(-y_{\mathrm{src}(\alpha)}), \tag{4}$$

where $n_0$ is the number of unconstrained variables in $\sigma$, and $n_*$ the number of variables taking *. Comparing RSP with SP-$\rho$ in [14], we see that

**Theorem 1.** *In the limit where all $y_a, y_i \to \infty$, RSP is equivalent to SP-$\rho$ [14], with $\rho = \omega_*$.*

Taking **y** to infinity correspond to disallowing violated constraints, and SP-$\rho$ was formulated for satisfiable SAT problems, where violated constraints are forbidden. In this case, all clauses must be satisfied and the term $\prod_{\alpha \in C} \exp(w_\alpha)$ in Equation 4 is a constant, and $P(\sigma) \propto \omega_0^{n_0} \omega_*^{n_*}$.

## 3.1 Main result

In the following, we assume the following settings: (1) $\omega_* = 1$ and $\omega_0 = 0$ ; (2) for positivity clauses $\beta(i)$, let $y_i = 0$ ; and (3) in the original MRF $G = (V, F)$, single-variable factors are defined on all variables (we can always define uniform factors). Under these settings, we will prove the main result that the joint distribution on the *relaxed* MRF is approximately equal to that on the original MRF, and that RSP estimates marginals on the original MRF. First, we prove the following lemma:

**Lemma 1.** *The joint probability over valid configurations on $G_s$ is proportional to the joint probability of the corresponding configurations on the original MRF, $G = (V, F)$.*

*Proof.* For valid configurations, all positivity clauses are satisfied, and for each $a \in F$, all valid configurations have one violated constraint in the set of interaction clauses $\{\alpha\}_{\alpha \sqsubset a}$. Hence the penalty term for violated constraints $\prod_{a \in F} \exp(y_a)$ is a constant factor. Let $W = \sum_{\alpha \in C} w_\alpha$ be the sum of all weights. For a valid configuration $\sigma$,

$$P(\sigma) \quad \propto \quad \exp(\sum_{\gamma \in \text{SAT}(\sigma)} w_\gamma) = \exp(W - \sum_{\gamma \in \text{UNSAT}(\sigma)} w_\gamma) \quad \propto \quad \prod_{a \in F} \Phi_a(x(\sigma))$$

$\square$

**Lemma 2.** *All configurations containing * have zero probability in the MRF $G_s$, and there is a one-to-one mapping between configurations $\lambda = \{\sigma_k, P_k\}_{k \in V_s}$ and configurations $\sigma = \{\sigma_k\}_{k \in B}$*

*Proof.* Single-variable factors on $G$ translate into single-literal clauses in the WMS formulation, which in turn becomes single-variable factors in $G_s$. For a variable $\lambda_k = (\sigma_k, P_k)$ with a single-variable factor, $\Psi_\alpha$, we have $\text{VAL}_\alpha(\sigma_k = *) = 0$. This implies $\Psi_\alpha(\lambda_k = (*, P_k)) = 0$. $\square$

**Lemma 3.** *MTO configurations have $n_0 \neq 0$ and since $\omega_0 = 0$, they have zero probability.*

*Proof.* In MTO configurations, $\exists (i, x_i, x_i'), \sigma_{i,x_i} = \sigma_{i,x_i'} = 1$. The positivity clause $\beta(i)$ is hence non-constraining for these variables, and since all other clauses connected to them are interaction clauses and contain them as negative literals, both variables are unconstrained. Hence $n_0 \neq 0$, and from Equation 4, for $\omega_0 = 0$, they have zero probability. $\square$

The above lemma lead to the following theorem:

**Theorem 2.** *Assuming that $\exp(w_{\beta(i)}) \gg 1$ for all $X_i \in V$, the joint distribution over the* relaxed *MRF $G_s = (V_s, F_s)$ is approximately equal to the joint distribution over the original MRF, $G = (V, F)$. Moreover, RSP estimates the marginals on the original MRF, and at the fixed points, the beliefs at each node, $B(\sigma_{(i,x_i)} = 1)$, is an estimate of $P(X_i = x_i)$, and $\sum_{x_i \in Q} B(\sigma_{(i,x_i)} = 1) \approx 1$.*

We can understand the above theorem as follows: if we assume that the probability of AZ invalid configurations is negligible (equivalent to assuming that the probability of violating positivity clauses are negligible, i.e. $\exp(w_i) \gg \exp(-y_{\text{src}(\beta(i))}) = 1$), then we have only valid configurations left. MTO invalid configurations are ruled out by Lemma 3. Since the positivity clauses have large weights, $\exp(w_i) \gg 1$ are usually satisfied. Hence RSP, as the sum-product algorithm on the *relaxed* MRF, returns estimations of the marginals $P(X_i = x_i)$ as $B(\sigma_{(i,x_i)} = 1)$.

## 3.2 Choosing y

Valid configurations have a joint probability with the factor $\prod_{a \in F} \exp(-y_a)$ while AZ configurations do not. However, Theorem 2 states that, if $\exp(w_i) \gg 1$, AZ configurations have negligible probability. Empirically, we observe that for a large range of values of $\{y_a\}_{a \in F}$, RSP returns marginals satisfying $\sum_{x_i} B(\sigma_{(i,x_i)} = 1) \approx 1$, indicating that AZ configurations do indeed have negligible probability. We can hence select the values of $\{y_a\}_{a \in F}$ for better convergence properties.

We describe heuristics based on the sufficient conditions for convergence of sum-product algorithms in [16]. To simplify notations, we write the conditions for a MRF with pairwise factors $\Phi_a$,

$$\max_{X_j \in V, b \in N(j)} \sum_{a \in N(j) \setminus b} \quad N(\Phi_a) < 1$$
$$\text{where} \qquad N(\Phi_a) = \sup_{x_i \neq x_i'} \sup_{x_j \neq x_j'} \tanh\left(\tfrac{1}{4} \log\left(\frac{\Phi_a(x_i,x_j)}{\Phi_a(x_i',x_j)} \frac{\Phi_a(x_i',x_j')}{\Phi_a(x_i,x_j')}\right)\right) \tag{5}$$

Mooij and Kappen [16] have also derived another condition based on the spectral radius of a matrix having $N(\Phi_a)$ as entries. These conditions lead us to believe that the sum-product algorithm converges better on MRFs with small $N(\Phi_a)$ (or the "strengths" of potentials in [8]). To calculate $N(\Psi_\alpha)$ for the interaction clause $\alpha$, we characterize these factors as follows:

$$\Psi_\alpha((\sigma_k, P_k),(\sigma_l, P_l)) = \begin{cases} \exp(-y_{\text{src}(\alpha)}) & \text{if clause } \alpha \text{ is violated, i.e. } (\sigma_k, \sigma_l) = (0,0) \\ \exp(w_\alpha) & \text{otherwise} \end{cases} \tag{6}$$

As $y_a$ are shared among $\alpha \sqsubset a$, we choose $y_a$ to minimize $\sum_{\alpha \sqsubset a} N(\Psi_\alpha) = \sum_{\alpha \sqsubset a} \tanh \frac{1}{4}|w_\alpha + y_a|$. A good approximation for $y_a$ would be the median of $\{-w_\alpha\}_{\alpha \sqsubset a}$. For our experiments, we divide the search range for $y_a$ into 10 bins, and use fminsearch in Matlab to find a local minimum.

### 3.3 Update equations and efficiency

While each message in RSP has large cardinality, we show in the supplementary material that, under the settings of Section 3.1, the update equations can be simplified such that each factor passes a single number to each variable. The interaction clause $\alpha$ sends a number $\nu_{\alpha \to (i,v)}$ to each $(i,x) \in C(\alpha)$, and the positivity clauses $\beta(i)$ sends a number $\mu_{\beta(i) \to (i,x)}$ to $(i,x)$ for each $x \in Q$. The update equations are as follows: (proofs in the supplementary material):

$$\mu_{\beta \to (i,x)} = \sum_{x' \neq x} \prod_{\alpha \in N(i,x') \setminus \beta(i)} \nu_{\alpha \to (i,x')} + \exp(-w_i) \tag{7}$$

$$\nu_{\alpha \to (i,x)} = \frac{\mu_{\beta(j) \to (j,x')} + \exp(-y_{\text{src}(\alpha)} - w_\alpha) \prod_{\gamma \in N(j,x') \setminus \{\beta(j),\alpha\}} \nu_{\gamma \to (j,x')}}{\mu_{\beta(j) \to (j,x')} + \prod_{\gamma \in N(j,x') \setminus \{\beta(j),\alpha\}} \nu_{\gamma \to (j,x')}} \tag{8}$$

$$B(\sigma_{(i,x)} = 0) \propto \mu_{\beta(i) \to (i,x)} \;\; ; \;\; B(\sigma_{(i,x)} = 1) \propto \prod_{\alpha \in N(i,x) \setminus \beta(i)} \nu_{\alpha \to (i,x)} \;\; ; \;\; B(\sigma_{(i,x)} = *) = 0 \tag{9}$$

We found empirically that the schedule of message updates affect convergence to a large extent. A good schedule is to update all the $\nu$-messages first (by updating the groups of $\nu$-messages belonging to each factor $a \in F$ together), and then updating the $\mu$-messages together. This seems to work better than the schedule defined by residual belief propagation [6] on the *relaxed* MRF.

In terms of efficiency, for a MRF with $N$ pairwise factors, the sum-product algorithm has $2Nq$ real numbers in the factor to variable messages, and RSP has $2Nq + q$. Empirically, we observe that RSP on the *relaxed* MRF runs as fast as the simple sum-product algorithm on the original MRF, with an overhead for determining the values of $\mathbf{y}$.

## 4 Experimental Results

While Ising models with attractive couplings are exactly solvable by graph-cut algorithms, general Ising models with mixed couplings on complete graphs are NP-hard [4], and graph cut algorithms are not applicable to graphs with mixed couplings [12]. In this section, we perform three sets of experiments to show that RSP outperforms other approaches: the first set compares RSP and the residual belief propagation on a simple graph, the second set compares the performance of various methods on randomly generated graphs with mixed couplings, and the third set applies RSP to the application of the web person disambiguation task.

**A simple example:** we use a 4-node complete graph of binary variables, with the two sets of factors defined in Figure 2(a), for $\epsilon = +1$ and -1. The case $\epsilon = -1$ was used in [8] to illustrate how the strengths of potentials affect convergence of the sum-product algorithm. We also show the case of $\epsilon = +1$ (an attractive network) as a case where the sum-product algorithm converges well. Both sets of graphs ($\epsilon = +1$ or $-1$) have uniform marginals, and 2 MAP configurations (modes). In Figure

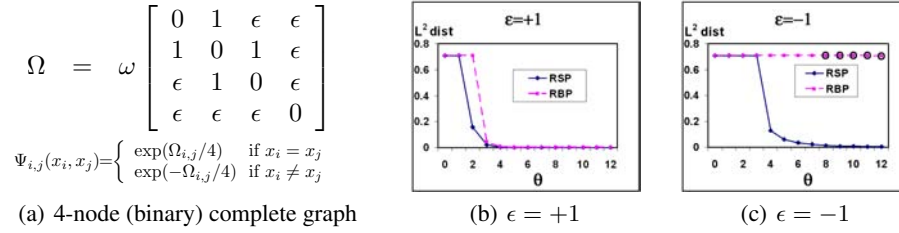

$$\Omega \;=\; \omega \begin{bmatrix} 0 & 1 & \epsilon & \epsilon \\ 1 & 0 & 1 & \epsilon \\ \epsilon & 1 & 0 & \epsilon \\ \epsilon & \epsilon & \epsilon & 0 \end{bmatrix}$$

$$\Psi_{i,j}(x_i,x_j)=\begin{cases} \exp(\Omega_{i,j}/4) & \text{if } x_i = x_j \\ \exp(-\Omega_{i,j}/4) & \text{if } x_i \neq x_j \end{cases}$$

(a) 4-node (binary) complete graph      (b) $\epsilon = +1$      (c) $\epsilon = -1$

Figure 2: In Figure (a), we define factors under the two settings: $\epsilon = \pm 1$. Figure (b) and (c) show the $L^2$ distance between the returned marginals and the nearest mode of the graph. Circles on the lines mean failure to converge, where we take the marginals at the last iteration.

2(b) and 2(c), we show experimental results for $\epsilon = +1$ and $-1$. In each case, we vary $\omega$ from 0 to 12, and for each $\omega$, run residual belief propagation (RBP) damped at 0.5 and RSP (undamped) on the corresponding graph. Both methods are randomly initialized. We plot the $L^2$ distance between the returned marginals and the nearest mode marginals (marginals with probability one on the modes). The correct marginals are uniform, where the $L^2$ distance is $\sqrt{0.5} \approx 0.7$. For small $\omega$, both methods converge to the correct marginals. As $\omega$ is increased, for $\epsilon = +1$ in Figure 2(b), both approaches converge to marginals with probability 1 on one of the modes. For $\epsilon = -1$, however, RSP converges again to marginals indicating a mode, while RBP faces convergence problems for $\omega \geq 8$.

Increasing $\omega$ corresponds to increasing $N(\Psi_{i,j})$, and the sum-product algorithm fails to converge for large $\omega$ when $\epsilon = -1$. When the algorithms converge for large $\omega$, they converge not to the correct marginals, but to a MAP configuration. Increasing $\omega$ has the same effect as decreasing the temperature of a network: the behavior of sum-product algorithm approaches that of the max-product algorithm, i.e. the max-product algorithm is the sum-product algorithm at the zero temperature limit.

**Ising models with mixed couplings:** we conduct experiments on complete graphs of size 20 with different percentage of attractive couplings, using the Ising model with the energy function: $H(s) = -\sum_{i,j} \theta_{i,j} s_i s_j - \sum_i \theta_i s_i$, where $s_i \in \{-1, 1\}$. We draw $\theta_i$ from $U[0, 0.1]$. To control the percentage of attractive couplings, we draw $\theta_{i,j}$ from $U[0, \alpha]$, and randomly assign negative signs to the $\theta_{i,j}$ with probability $(1 - \rho)$, where $\rho$ is the percentage of attractive couplings required. We vary $\alpha$ from 1 to 3. In Figure 3, we plot the difference between the optimal energy (obtained with a brute force search) and the energy returned by each of the following approaches: RSP, max-product belief propagation (MBP), the convergent tree reweighted max product belief propagation (TRW-S) [11], residual sum-product belief propagation (RBP) [6], and tree-structured expecation propagation (TEP) [15]. Each point on the graph is the average over 30 randomly generated networks. In Table 1, we compare RSP against these methods. When an algorithm does not converge, we take its result at the last iteration. We damp RBP and TEP with a 0.5 damping factor. For RSP, MBP, TRW-S and RBP, we randomly initialize the initial messages, and take the best result after 5 restarts. For TEP, we use five different trees consisting of a maximal spanning tree and four random stars [19]. For RSP, RBP and TEP, which are variants of the sum product algorithm, we lower the temperature by a factor of 2 each time the method converges and stop when the method fails to converge or if the results are not improved over the last temperature. We observe that MBP outperforms TRW-S constantly: this agrees with [11] that MBP outperforms TRW-S for graphs with mixed couplings. While the performance of TRW-S remains constant from 25% to 75%, the sum-product based algorithms (RBP and TEP) improve as the percentage of attractive potentials is increased. In all three cases, RSP is one of the best performing methods, beaten only by TEP at 2 points on the 50% graph. TEP, being of the class of generalized belief propagation [19], runs significantly slower than RSP.

**Supervised clustering:** Finley and Joachims [7] formulated $SVM^{cluster}$, which learns an item-pair similarity measure, $\text{Sim}(i, j)$, to minimize a correlation clustering objective on a training set. In training $SVM^{cluster}$, they have to minimize $E(\mathbf{x}) = \sum_{i,j} \text{Sim}(i, j)\delta(x_i, x_j)$ where $x_i \in \{1, .., U\}$ are cluster-ids of item $i$, and $U$ an upper-bound on the number of clusters. They tried a greedy and a linear programming approach, and concluded that the two approaches are comparable.

Due to time constraints, we did not implement $SVM^{cluster}$: instead we test our inference algorithms on the pairwise classification clustering (PCC) baseline in [7]. The PCC baseline trains *svmlight* [9] on training item-pairs, and run the classifier through all test pairs. For each test pair $(i, j)$, we apply softmax to the classifier outputs to obtain the probability $p_{i,j}$ that the pair is in the same cluster.

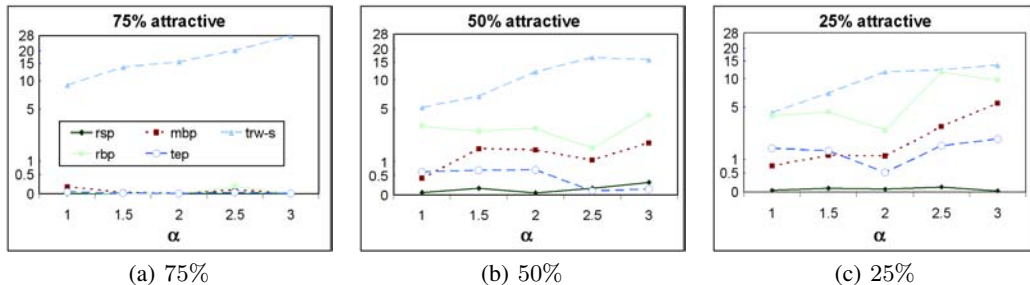

(a) 75%    (b) 50%    (c) 25%

Figure 3: Experiments on the complete graph Ising model with mixed couplings (legend in (a)), with different percentage of attractive couplings. The y-axis shows, in log scale, the average energy difference between the configuration found by the algorithm and the optimal solution.

| | 75% attractive | | | | | 50% attractive | | | | | 25% attractive | | | | |
|---|---|---|---|---|---|---|---|---|---|---|---|---|---|---|---|
| $\alpha$ | 1 | 1.5 | 2 | 2.5 | 3 | 1 | 1.5 | 2 | 2.5 | 3 | 1 | 1.5 | 2 | 2.5 | 3 |
| mbp | 2/0 | 2/0 | 0/0 | 1/0 | 1/0 | 7/6 | 11/5 | 14/0 | 10/2 | 9/6 | 20/2 | 13/3 | 16/0 | 13/3 | 15/2 |
| trw-s | 26/0 | 24/0 | 22/0 | 25/0 | 25/0 | 28/0 | 29/0 | 29/0 | 27/0 | 28/1 | 29/0 | 27/0 | 30/0 | 28/1 | 27/0 |
| rbp | 1/0 | 0/0 | 0/0 | 2/0 | 0/0 | 22/0 | 14/2 | 12/0 | 9/1 | 13/5 | 22/0 | 16/6 | 15/2 | 21/0 | 17/0 |
| tep | 2/0 | 2/0 | 0/0 | 2/0 | 0/0 | 14/3 | 9/3 | 11/2 | 6/2 | 6/5 | 23/1 | 15/4 | 10/2 | 16/2 | 15/2 |
| opt | 0/0 | 0/0 | 0/0 | 0/1 | 0/0 | 0/7 | 0/8 | 0/2 | 0/2 | 0/7 | 0/6 | 0/10 | 0/4 | 0/4 | 0/2 |

Table 1: Number of trials (out of 30) where RSP does better/worse than various methods. In particular, the last row (opt) shows the number of times that RSP does worse than the optimal solution.

Defining $\mathrm{Sim}(i,j) = \log(p_{i,j}/(1-p_{i,j}))$, we minimize $E(\mathbf{x})$ to cluster the test set. We found that the various inference algorithms perform poorly on the MRF for large $U$, even when they converge (probably due to a large number of minima in the approximation). We are able to obtain lower energy configurations by the recursive 2-way partitioning procedure in [5] used for graph cuts. (Graph cuts do not apply here as weights can be negative). This procedure involves recursively running, for e.g. RSP, on the MRF for $E(\mathbf{x})$ with $U = 2$, and applying the Kernighan-Lin algorithm [10] for local refinements among current partitions. Each time RSP returns a configuration that partitions the data, we run RSP on each of the two partitions. We terminate the recursion when RSP assigns a same value to all variables, placing all remaining items in one cluster.

We use the web person disambiguation task defined in SemEval-2007 [1] as the test application. Training data consists of 49 sets of web pages (we use 29 sets with more than 50 documents), where each set (or domain) are results from a search query on a person name. The test data contains another 30 domains. Each domain is manually annotated into clusters, with each cluster containing pages referring to a single individual. We use a simple feature filtering approach to select features that are useful across many domains in the training data. Candidate features include (i) words occurring in only one document of the document-pair, (ii) words co-ocurring in both documents, (iii) named entity matches between the documents, and (iv) topic correlation features. For comparison, we replace RSP with MBP and TRW-S as inference algorithms (we did not run RBP and TEP as they are very slow on these problems because they often fail to converge). We also implemented the greedy algorithm (Greedy) in [7]. We tried using the linear programming approach but free off-the-shelf solvers seem unable to scale to these problems. Results comparing RSP with Greedy, MBP and TRW-S are shown in Table 2. The F-measure attained by RSP for this SemEval task [1] is equal to the systems ranked second and third out of 16 participants (official results yet unpublished). We found that although TRW-S is guaranteed to converge, it performs poorly. RSP converges far better than MBP, but due to the Kernighan-Lin corrections that we run at each iteration, results can sometimes be corrected to a large extent by the local refinements.

| *Method* | RSP | MBP | TRW-S | Greedy |
|---|---|---|---|---|
| Number of test domains where RSP attains lower/higher energy $E(\mathbf{x})$ than *Method* | 0/0 | 9/6 | 16/7 | 22/5 |
| Percentage of convergence over all runs | 91% | 74% | 100% * | - |
| F-measure of purity and inverse purity [1] | 75.08% | 74.97% | 74.61% | 74.78% |

Table 2: Results for the web person disambiguation task. (*: TRW-S is guaranteed to converge)

## 5  Related work and conclusion

In this paper, we formulated RSP, generalizing the formulation of SP-$\rho$ in [14]. SP-$\rho$ is the sum-product interpretation for the survey propagation (SP) algorithm [3]. SP has been shown to work well

for hard instances of 3-SAT, near the phase transition where local search algorithms fail. However, its application has been limited to constraint satisfaction problems [3]. In RSP, we took inspiration from the SP-$y$ algorithm [2] in adding a penalty term for violated clauses. SP-$y$ works on MAX-SAT problems and SP can be considered as SP-$y$ with $y$ taken to $\infty$, hence disallowing violated constraints. This is analogous to the relation between RSP and SP-$\rho$ [14] (See Theorem 1). RSP is however different from SP-$y$ since we address weighted MAX-SAT problems. Even if all weights are equal, RSP is still different from SP-$y$, which, so far, does not have a sum-product formulation on an alternative MRF. We show that while RSP is the sum-product algorithm on a relaxed MRF, it can be used for solving the energy minimization problem. By tuning the strengths of the factors (based on convergence criteria in [16]) while keeping the underlying distribution approximately correct, RSP converges well even at low temperatures. This enables it to return low-energy configurations on MRFs where other methods fail. As far as we know, this is the first application of convergence criteria to aid convergence of belief propagation algorithms, and this mechanism can be used to exploit future work on sufficient conditions for the convergence of belief propagation algorithms.

**Acknowledgments**
We would like to thank Yee Fan Tan for his help on the web person disambiguation task, and Tomas Lozano-Perez and Leslie Pack Kaelbling for valuable comments on the paper. The research is partially supported by ARF grant R-252-000-240-112.

# References

[1] "Web person disambiguation task at SemEval," 2007. [Online]. Available: http://nlp.uned.es/weps/task-description-2.html

[2] D. Battaglia, M. Kolar, and R. Zecchina, "Minimizing energy below the glass thresholds," *Physical Review E*, vol. 70, 2004.

[3] A. Braunstein, M. Mezard, and R. Zecchina, "Survey propagation: An algorithm for satisfiability," *Random Struct. Algorithms*, vol. 27, no. 2, 2005.

[4] B. A. Cipra, "The Ising model is NP-complete," *SIAM News*, vol. 33, no. 6, 2000.

[5] C. Ding, "Spectral clustering," ICML '04 Tutorial, 2004.

[6] G. Elidan, I. McGraw, and D. Koller, "Residual belief propagation: Informed scheduling for asynchronous message passing," in *UAI*, 2006.

[7] T. Finley and T. Joachims, "Supervised clustering with support vector machines," in *ICML*, 2005.

[8] T. Heskes, "On the uniqueness of loopy belief propagation fixed points," *Neural Computation*, vol. 16, 2004.

[9] T. Joachims, *Learning to Classify Text Using Support Vector Machines: Methods, Theory and Algorithms.* Norwell, MA, USA: Kluwer Academic Publishers, 2002.

[10] B. Kernighan and S. Lin, "An efficient heuristic procedure for partitioning graphs," Bell Systems Technical Report, 1970.

[11] V. Kolmogorov, "Convergent tree-reweighted message passing for energy minimization," *IEEE Transactions on Pattern Analysis and Machine Intelligence*, vol. 28, no. 10, 2006.

[12] V. Kolmogorov and R. Zabih, "What energy functions can be minimized via graph cuts?" *IEEE Transactions on Pattern Analysis and Machine Intelligence*, vol. 26, no. 2, 2004.

[13] F. Kschischang, B. Frey, and H. Loeliger, "Factor graphs and the sum-product algorithm," *IEEE Transactions on Information Theory*, vol. 47, no. 2, 2001.

[14] E. Maneva, E. Mossel, and M. Wainright, "A new look at survey propagation and its generalizations," 2004. [Online]. Available: http://arxiv.org/abs/cs.CC/0409012

[15] T. Minka and Y. Qi, "Tree-structured approximations by expectation propagation," in *NIPS*, 2004.

[16] J. M. Mooij and H. J. Kappen, "Sufficient conditions for convergence of loopy belief propagation," in *UAI*, 2005.

[17] J. D. Park, "Using weighted MAX-SAT engines to solve MPE," in *AAAI*, 2002.

[18] Y. Weiss and W. T. Freeman, "On the optimality of solutions of the max-product belief-propagation algorithm in arbitrary graphs," *IEEE Transactions on Information Theory*, vol. 47, no. 2, 2001.

[19] M. Welling, T. Minka, and Y. W. Teh, "Structured region graphs: Morphing EP into GBP," in *UAI*, 2005.

[20] J. S. Yedidia, W. T. Freeman, and Y. Weiss, "Constructing free-energy approximations and generalized belief propagation algorithms," *IEEE Transactions on Information Theory*, vol. 51, no. 7, 2005.
